# Switched Latent Force Models
# for Movement Segmentation

**Mauricio A. Álvarez** [1]**, Jan Peters** [2]**, Bernhard Schölkopf** [2]**, Neil D. Lawrence** [3,4]
[1] School of Computer Science, University of Manchester, Manchester, UK M13 9PL
[2] Max Planck Institute for Biological Cybernetics, Tübingen, Germany 72076
[3] School of Computer Science, University of Sheffield, Sheffield, UK S1 4DP
[4] The Sheffield Institute for Translational Neuroscience, Sheffield, UK S10 2HQ

## Abstract

Latent force models encode the interaction between multiple related dynamical systems in the form of a kernel or covariance function. Each variable to be modeled is represented as the output of a differential equation and each differential equation is driven by a weighted sum of latent functions with uncertainty given by a Gaussian process prior. In this paper we consider employing the latent force model framework for the problem of determining robot motor primitives. To deal with discontinuities in the dynamical systems or the latent driving force we introduce an extension of the basic latent force model, that switches between different latent functions and potentially different dynamical systems. This creates a versatile representation for robot movements that can capture discrete changes and non-linearities in the dynamics. We give illustrative examples on both synthetic data and for striking movements recorded using a Barrett WAM robot as haptic input device. Our inspiration is robot motor primitives, but we expect our model to have wide application for dynamical systems including models for human motion capture data and systems biology.

## 1  Introduction

Latent force models [1] are a new approach for modeling data that allows combining dimensionality reduction with systems of differential equations. The basic idea is to assume an observed set of $D$ correlated functions to arise from an unobserved set of $R$ forcing functions. The assumption is that the $R$ forcing functions drive the $D$ observed functions through a set of differential equation models. Each differential equation is driven by a weighted mix of latent forcing functions. Sets of coupled differential equations arise in many physics and engineering problems particularly when the temporal evolution of a system needs to be described. Learning such differential equations has important applications, e.g., in the study of human motor control and in robotics [6]. A latent force model differs from classical approaches as it places a probabilistic process prior over the latent functions and hence can make statements about the uncertainty in the system. A joint Gaussian process model over the latent forcing functions and the observed data functions can be recovered using a Gaussian process prior in conjunction with linear differential equations [1]. The resulting latent force modeling framework allows the combination of the knowledge of the systems dynamics with a data driven model. Such generative models can be used to good effect, for example in ranked target prediction for transcription factors [5].

If a single Gaussian process prior is used to represent each latent function then the models we consider are limited to smooth driving functions. However, discontinuities and segmented latent forces are omnipresent in real-world data. For example, impact forces due to contacts in a mechanical dynamical system (when grasping an object or when the feet touch the ground) or a switch in an electrical circuit result in discontinuous latent forces. Similarly, most non-rhythmic natural mo-

tor skills consist of a sequence of segmented, discrete movements. If these segments are separate time-series, they should be treated as such and *not* be modeled by the same Gaussian process model.

In this paper, we extract a sequence of dynamical systems motor primitives modeled by second order linear differential equations in conjunction with forcing functions (as in [1, 6]) from human movement to be used as demonstrations of elementary movements for an anthropomorphic robot. As human trajectories have a large variability: both due to planned uncertainty of the human's movement policy, as well as due to motor execution errors [7], a probabilistic model is needed to capture the underlying motor primitives. A set of second order differential equations is employed as mechanical systems are of the same type and a temporal Gaussian process prior is used to allow probabilistic modeling [1]. To be able to obtain a sequence of dynamical systems, we augment the latent force model to include discontinuities in the latent function and change dynamics. We introduce discontinuities by switching between different Gaussian process models (superficially similar to a mixture of Gaussian processes; however, the switching times are modeled as parameters so that at any instant a single Gaussian process is driving the system). Continuity of the observed functions is then ensured by constraining the relevant state variables (for example in a second order differential equation velocity and displacement) to be continuous across the switching points. This allows us to model highly non stationary multivariate time series. We demonstrate our approach on synthetic data and real world movement data.

## 2   Review of Latent force models (LFM)

Latent force models [1] are hybrid models that combine mechanistic principles and Gaussian processes as a flexible way to introduce prior knowledge for data modeling. A set of $D$ functions $\{y_d(t)\}_{d=1}^D$ is modeled as the set of output functions of a series of coupled differential equations, whose common input is a linear combination of $R$ latent functions, $\{u_r(t)\}_{r=1}^R$. Here we focus on a second order ordinary differential equation (ODE). We assume the output $y_d(t)$ is described by

$$A_d \frac{\mathrm{d}^2 y_d(t)}{\mathrm{d}t^2} + C_d \frac{\mathrm{d}y_d(t)}{\mathrm{d}t} + \kappa_d y_d(t) = \sum_{r=1}^R S_{d,r} u_r(t),$$

where, for a mass-spring-damper system, $A_d$ would represent the mass, $C_d$ the damper and $\kappa_d$, the spring constant associated to the output $d$. We refer to the variables $S_{d,r}$ as the sensitivity parameters. They are used to represent the relative strength that the latent force $r$ exerts over the output $d$. For simplicity we now focus on the case where $R = 1$, although our derivations apply more generally. Note that models that learn a forcing function to drive a linear system have proven to be well-suited for imitation learning for robot systems [6]. The solution of the second order ODE follows

$$y_d(t) = y_d(0)c_d(t) + \dot{y}_d(0)e_d(t) + f_d(t, u), \tag{1}$$

where $y_d(0)$ and $\dot{y}_d(0)$ are the output and the velocity at time $t = 0$, respectively, known as the initial conditions (IC). The angular frequency is given by $\omega_d = \sqrt{(4A_d \kappa_d - C_d^2)/(4A_d^2)}$ and the remaining variables are given by

$$c_d(t) = e^{-\alpha_d t}\left[\cos(\omega_d t) + \frac{\alpha_d}{\omega_d}\sin(\omega_d t)\right], \quad e_d(t) = \frac{e^{-\alpha_d t}}{\omega_d}\sin(\omega_d t),$$

$$f_d(t, u) = \frac{S_d}{A_d \omega_d}\int_0^t G_d(t - \tau)u(\tau)\mathrm{d}\tau = \frac{S_d}{A_d \omega_d}\int_0^t e^{-\alpha_d(t-\tau)}\sin[(t - \tau)\omega_d]u(\tau)\mathrm{d}\tau,$$

with $\alpha_d = C_d/(2A_d)$. Note that $f_d(t, u)$ has an implicit dependence on the latent function $u(t)$. The uncertainty in the model of Eq. (1) is due to the fact that the latent force $u(t)$ and the initial conditions $y_d(0)$ and $\dot{y}_d(0)$ are not known. We will assume that the latent function $u(t)$ is sampled from a zero mean Gaussian process prior, $u(t) \sim \mathcal{GP}(0, k_{u,u}(t, t'))$, with covariance function $k_{u,u}(t, t')$.

If the initial conditions, $\mathbf{y}_{IC} = [y_1(0), y_2(0), \ldots, y_D(0), v_1(0), v_2(0), \ldots, v_D(0)]^\top$, are independent of $u(t)$ and distributed as a zero mean Gaussian with covariance $K_{IC}$ the covariance function between any two output functions, $d$ and $d'$ at any two times, $t$ and $t'$, $k_{y_d, y_{d'}}(t, t')$ is given by

$$c_d(t)c_{d'}(t')\sigma_{y_d, y_{d'}} + c_d(t)e_{d'}(t')\sigma_{y_d, v_{d'}} + e_d(t)c_{d'}(t')\sigma_{v_d, y_{d'}} + e_d(t)e_{d'}(t')\sigma_{v_d, v_{d'}} + k_{f_d, f_{d'}}(t, t'),$$

where $\sigma_{y_d, y_{d'}}$, $\sigma_{y_d, v_{d'}}$, $\sigma_{v_d, y_{d'}}$ and $\sigma_{v_d, v_{d'}}$ are entries of the covariance matrix $K_{IC}$ and

$$k_{f_d, f_{d'}}(t, t') = K_0 \int_0^t G_d(t - \tau) \int_0^{t'} G_{d'}(t' - \tau') k_{u,u}(t, t')\mathrm{d}\tau'\mathrm{d}\tau, \tag{2}$$

where $K_0 = S_d S_{d'} / (A_d A_{d'} \omega_d \omega_{d'})$. So the covariance function $k_{f_d, f_{d'}}(t, t')$ depends on the covariance function of the latent force $u(t)$. If we assume the latent function has a radial basis function (RBF) covariance, $k_{u,u}(t, t') = \exp[-(t - t')^2/\ell^2]$, then $k_{f_d, f_{d'}}(t, t')$ can be computed analytically [1] (see also supplementary material). The latent force model induces a joint Gaussian process model across all the outputs. The parameters of the covariance function are given by the parameters of the differential equations and the length scale of the latent force. Given a multivariate time series data set these parameters may be determined by maximum likelihood.

The model can be thought of as a set of mass-spring-dampers being driven by a function sampled from a Gaussian process. In this paper we look to extend the framework to the case where there can be discontinuities in the latent functions. We do this through switching between different Gaussian process models to drive the system.

## 3 Switching dynamical latent force models (SDLFM)

We now consider switching the system between different latent forces. This allows us to change the dynamical system and the driving force for each segment. By constraining the displacement and velocity at each switching time to be the same, the output functions remain continuous.

### 3.1 Definition of the model

We assume that the input space is divided in a series of non-overlapping intervals $[t_{q-1}, t_q]_{q=1}^Q$. During each interval, only one force $u_{q-1}(t)$ out of $Q$ forces is active, that is, there are $\{u_{q-1}(t)\}_{q=1}^Q$ forces. The force $u_{q-1}(t)$ is activated after time $t_{q-1}$ (switched on) and deactivated (switched off) after time $t_q$. We can use the basic model in equation (1) to describe the contribution to the output due to the sequential activation of these forces. A particular output $z_d(t)$ at a particular time instant $t$, in the interval $(t_{q-1}, t_q)$, is expressed as

$$z_d(t) = y_d^q(t - t_{q-1}) = c_d^q(t - t_{q-1})y_d^q(t_{q-1}) + e_d^q(t - t_{q-1})\dot{y}_d^q(t_{q-1}) + f_d^q(t - t_{q-1}, u_{q-1}).$$

This equation is assummed to be valid for describing the output only inside the interval $(t_{q-1}, t_q)$. Here we highlighted this idea by including the superscript $q$ in $y_d^q(t - t_{q-1})$ to represent the interval $q$ for which the equation holds, although later we will omit it to keep the notation uncluttered. Note that for $Q = 1$ and $t_0 = 0$, we recover the original latent force model given in equation (1). We also define the velocity $\dot{z}_d(t)$ at each time interval $(t_{q-1}, t_q)$ as

$$\dot{z}_d(t) = \dot{y}_d^q(t - t_{q-1}) = g_d^q(t - t_{q-1})y_d^q(t_{q-1}) + h_d^q(t - t_{q-1})\dot{y}_d^q(t_{q-1}) + m_d^q(t - t_{q-1}, u_{q-1}),$$

where $g_d(t) = -e^{-\alpha_d t}\sin(\omega_d t)(\alpha_d^2\omega_d^{-1} + \omega_d)$ and

$$h_d(t) = -e^{-\alpha_d t}\left[\frac{\alpha_d}{\omega_d}\sin(\omega_d t) - \cos(\omega_d t)\right], \quad m_d(t) = \frac{S_d}{A_d\omega_d}\frac{\mathrm{d}}{\mathrm{d}t}\left(\int_0^t G_d(t - \tau)u(\tau)\mathrm{d}\tau\right).$$

Given the parameters $\boldsymbol{\theta} = \{\{A_d, C_d, \kappa_d, S_d\}_{d=1}^D, \{\ell_{q-1}\}_{q=1}^Q\}$, the uncertainty in the outputs is induced by the prior over the initial conditions $y_d^q(t_{q-1}), \dot{y}_d^q(t_{q-1})$ for all values of $t_{q-1}$ and the prior over latent force $u_{q-1}(t)$ that is active during $(t_{q-1}, t_q)$. We place independent Gaussian process priors over each of these latent forces $u_{q-1}(t)$, assuming independence between them.

For initial conditions $y_d^q(t_{q-1}), \dot{y}_d^q(t_{q-1})$, we could assume that they are either parameters to be estimated or random variables with uncertainty governed by independent Gaussian distributions with covariance matrices $K_{IC}^q$ as described in the last section. However, for the class of applications we will consider: mechanical systems, the outputs should be continuous across the switching points. We therefore assume that the uncertainty about the initial conditions for the interval $q$, $y_d^q(t_{q-1}), \dot{y}_d^q(t_{q-1})$ are proscribed by the Gaussian process that describes the outputs $z_d(t)$ and velocities $\dot{z}_d(t)$ in the previous interval $q - 1$. In particular, we assume $y_d^q(t_{q-1}), \dot{y}_d^q(t_{q-1})$ are Gaussian-distributed with mean values given by $y_d^{q-1}(t_{q-1} - t_{q-2})$ and $\dot{y}_d^{q-1}(t_{q-1} - t_{q-2})$ and covariances $k_{z_d, z_{d'}}(t_{q-1}, t_{q'-1}) = \mathrm{cov}[y_d^{q-1}(t_{q-1} - t_{q-2}), y_{d'}^{q-1}(t_{q-1} - t_{q-2})]$ and $k_{\dot{z}_d, \dot{z}_{d'}}(t_{q-1}, t_{q'-1}) = \mathrm{cov}[\dot{y}_d^{q-1}(t_{q-1} - t_{q-2}), \dot{y}_{d'}^{q-1}(t_{q-1} - t_{q-2})]$. We also consider covariances between $z_d(t_{q-1})$ and $\dot{z}_{d'}(t_{q'-1})$, this is, between positions and velocities for different values of $q$ and $d$.

**Example 1.** Let us assume we have one output ($D = 1$) and three switching intervals ($Q = 3$) with switching points $t_0, t_1$ and $t_2$. At $t_0$, we assume that $\mathbf{y}_{IC}$ follows a Gaussian distribution with

mean zero and covariance $K_{IC}$. From $t_0$ to $t_1$, the output $z(t)$ is described by

$$z(t) = y^1(t - t_0) = c^1(t - t_0)y^1(t_0) + e^1(t - t_0)\dot{y}^1(t_0) + f^1(t - t_0, u_0).$$

The initial condition for the position in the interval $(t_1, t_2)$ is given by the last equation evaluated a $t_1$, this is, $z(t_1) = y^2(t_1) = y^1(t_1 - t_0)$. A similar analysis is used to obtain the initial condition associated to the velocity, $\dot{z}(t_1) = \dot{y}^2(t_1) = \dot{y}^1(t_1 - t_0)$. Then, from $t_1$ to $t_2$, the output $z(t)$ is

$$\begin{aligned} z(t) = y^2(t - t_1) &= c^2(t - t_1)y^2(t_1) + e^2(t - t_1)\dot{y}^2(t_1) + f^2(t - t_1, u_1), \\ &= c^2(t - t_1)y^1(t_1 - t_0) + e^2(t - t_1)\dot{y}^1(t_1 - t_0) + f^2(t - t_1, u_1). \end{aligned}$$

Following the same train of thought, the output $z(t)$ from $t_2$ is given as

$$z(t) = y^3(t - t_2) = c^3(t - t_2)y^3(t_2) + e^3(t - t_2)\dot{y}^3(t_2) + f^3(t - t_2, u_2),$$

where $y^3(t_2) = y^2(t_2 - t_1)$ and $\dot{y}^3(t_2) = \dot{y}^2(t_2 - t_1)$. Figure 1 shows an example of the switching dynamical latent force model scenario. To ensure the continuity of the outputs, the initial condition is forced to be equal to the output of the last interval evaluated at the switching point.

## 3.2 The covariance function

The derivation of the covariance function for the switching model is rather involved. For continuous output signals, we must take into account constraints at each switching time. This causes initial conditions for each interval to be dependent on final conditions for the previous interval and induces correlations across the intervals. This effort is worthwhile though as the resulting model is very flexible and can take advantage of the switching dynamics to represent a range of signals.

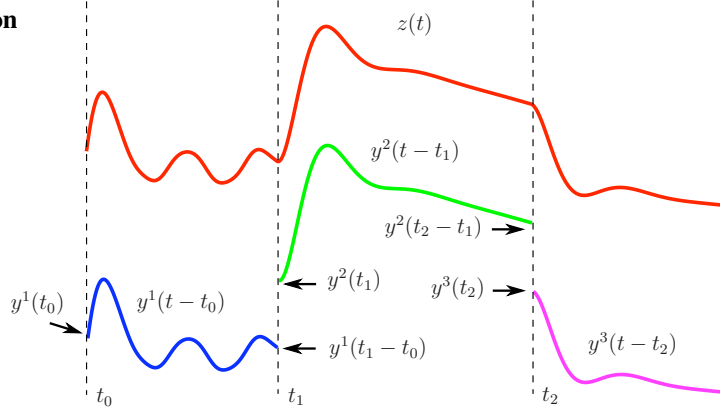

Figure 1: Representation of an output constructed through a switching dynamical latent force model with $Q = 3$. The initial conditions $y^q(t_{q-1})$ for each interval are matched to the value of the output in the last interval, evaluated at the switching point $t_{q-1}$, this is, $y^q(t_{q-1}) = y^{q-1}(t_{q-1} - t_{q-2})$.

As a taster, Figure 2 shows samples from a covariance function of a switching dynamical latent force model with $D = 1$ and $Q = 3$. Note that while the latent forces (a and c) are discrete, the outputs (b and d) are continuous and have matching gradients at the switching points. The outputs are highly nonstationary. The switching times turn out to be parameters of the covariance function. They can be optimized along with the dynamical system parameters to match the location of the nonstationarities. We now give an overview of the covariance function derivation. Details are provided in the supplementary material.

(a) System 1. Samples from the latent force.  (b) System 1. Samples from the output.  (c) System 2. Samples from the latent force.  (d) System 2. Samples from the output.

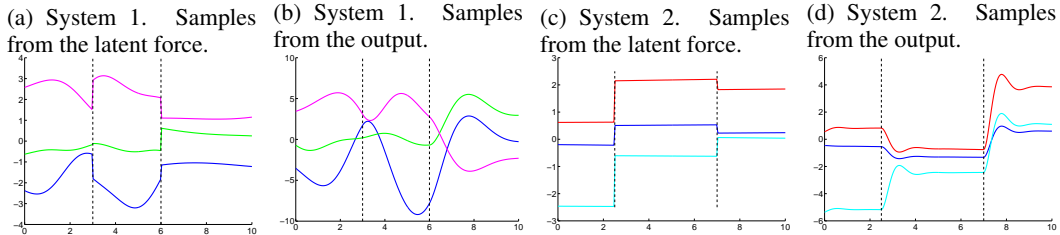

Figure 2: Joint samples of a switching dynamical LFM model with one output, $D = 1$, and three intervals, $Q = 3$, for two different systems. Dashed lines indicate the presence of switching points. While system 2 responds instantaneously to the input force, system 1 delays its reaction due to larger inertia.

In general, we need to compute the covariance $k_{z_d, z_{d'}}(t, t') = \text{cov}[z_d(t), z_{d'}(t')]$ for $z_d(t)$ in time interval $(t_{q-1}, t_q)$ and $z_{d'}(t')$ in time interval $(t_{q'-1}, t_{q'})$. By definition, this covariance follows

$$\text{cov}[z_d(t), z_{d'}(t')] = \text{cov}\left[y_d^q(t - t_{q-1}), y_{d'}^{q'}(t - t_{q'-1}))\right].$$

We assumme independence between the latent forces $u_q(t)$ and independence between the initial conditions $\mathbf{y}_{IC}$ and the latent forces $u_q(t)$.[1] With these conditions, it can be shown[2] that the covariance function[3] for $q = q'$ is given as

$$
\begin{aligned}
&c_d^q(t - t_{q-1})c_{d'}^q(t' - t_{q-1})k_{z_d, z_{d'}}(t_{q-1}, t_{q-1}) + c_d^q(t - t_{q-1})e_{d'}^q(t' - t_{q-1})k_{z_d, \dot{z}_{d'}}(t_{q-1}, t_{q-1}) \\
&+e_d^q(t - t_{q-1})c_{d'}^q(t' - t_{q-1})k_{\dot{z}_d, z_{d'}}(t_{q-1}, t_{q-1}) + e_d^q(t - t_{q-1})e_{d'}^q(t' - t_{q-1})k_{\dot{z}_d, \dot{z}_{d'}}(t_{q-1}, t_{q-1}) \\
&\hspace{6cm} +k_{f_d, f_{d'}}^q(t, t'),
\end{aligned}
\tag{3}
$$

where
$$k_{z_d, z_{d'}}(t_{q-1}, t_{q-1}) = \text{cov}[y_d^q(t_{q-1})y_{d'}^q(t_{q-1})], \quad k_{z_d, \dot{z}_{d'}}(t_{q-1}, t_{q-1}) = \text{cov}[y_d^q(t_{q-1})\dot{y}_{d'}^q(t_{q-1})],$$
$$k_{\dot{z}_d, z_{d'}}(t_{q-1}, t_{q-1}) = \text{cov}[\dot{y}_d^q(t_{q-1})y_{d'}^q(t_{q-1})], \quad k_{\dot{z}_d, \dot{z}_{d'}}(t_{q-1}, t_{q-1}) = \text{cov}[\dot{y}_d^q(t_{q-1})\dot{y}_{d'}^q(t_{q-1})].$$
$$k_{f_d, f_{d'}}^q(t, t') = \text{cov}[f_d^q(t - t_{q-1})f_{d'}^q(t' - t_{q-1})].$$

In expression (3), $k_{z_d, z_{d'}}(t_{q-1}, t_{q-1}) = \text{cov}[y_d^{q-1}(t_{q-1} - t_{q-2}), y_{d'}^{q-1}(t_{q-1} - t_{q-2})]$ and values for $k_{z_d, \dot{z}_{d'}}(t_{q-1}, t_{q-1})$, $k_{\dot{z}_d, z_{d'}}(t_{q-1}, t_{q-1})$ and $k_{\dot{z}_d, \dot{z}_{d'}}(t_{q-1}, t_{q-1})$ can be obtained by similar expressions. The covariance $k_{f_d, f_{d'}}^q(t, t')$ follows a similar expression that the one for $k_{f_d, f_{d'}}(t, t')$ in equation (2), now depending on the covariance $k_{u_{q-1}, u_{q-1}}(t, t')$. We will assume that the covariances for the latent forces follow the RBF form, with length-scale $\ell_q$.

When $q > q'$, we have to take into account the correlation between the initial conditions $y_d^q(t_{q-1})$, $\dot{y}_d^q(t_{q-1})$ and the latent force $u_{q'-1}(t')$. This correlation appears because of the contribution of $u_{q'-1}(t')$ to the generation of the initial conditions, $y_d^q(t_{q-1})$, $\dot{y}_d^q(t_{q-1})$. It can be shown[4] that the covariance function $\text{cov}[z_d(t), z_{d'}(t')]$ for $q > q'$ follows

$$
\begin{aligned}
&c_d^q(t - t_{q-1})c_{d'}^{q'}(t' - t_{q'-1})k_{z_d, z_{d'}}(t_{q-1}, t_{q'-1}) + c_d^q(t - t_{q-1})e_{d'}^{q'}(t' - t_{q'-1})k_{z_d, \dot{z}_{d'}}(t_{q-1}, t_{q'-1}) \\
&+e_d^q(t - t_{q-1})c_{d'}^{q'}(t' - t_{q'-1})k_{\dot{z}_d, z_{d'}}(t_{q-1}, t_{q'-1}) + e_d^q(t - t_{q-1})e_{d'}^{q'}(t' - t_{q'-1})k_{\dot{z}_d, \dot{z}_{d'}}(t_{q-1}, t_{q'-1}) \\
&\hspace{1cm} +c_d^q(t - t_{q-1})\mathcal{X}_d^1 k_{f_d, f_{d'}}^{q'}(t_{q'-1}, t') + c_d^q(t - t_{q-1})\mathcal{X}_d^2 k_{m_d, f_{d'}}^{q'}(t_{q'-1}, t') \\
&\hspace{1cm} +e_d^q(t - t_{q-1})\mathcal{X}_d^3 k_{f_d, f_{d'}}^{q'}(t_{q'-1}, t') + e_d^q(t - t_{q-1})\mathcal{X}_d^4 k_{m_d, f_{d'}}^{q'}(t_{q'-1}, t'),
\end{aligned}
\tag{4}
$$

where
$$k_{z_d, z_{d'}}(t_{q-1}, t_{q'-1}) = \text{cov}[y_d^q(t_{q-1})y_{d'}^{q'}(t_{q'-1})], \quad k_{z_d, \dot{z}_{d'}}(t_{q-1}, t_{q'-1}) = \text{cov}[y_d^q(t_{q-1})\dot{y}_{d'}^{q'}(t_{q'-1})],$$
$$k_{\dot{z}_d, z_{d'}}(t_{q-1}, t_{q'-1}) = \text{cov}[\dot{y}_d^q(t_{q-1})y_{d'}^{q'}(t_{q'-1})], \quad k_{\dot{z}_d, \dot{z}_{d'}}(t_{q-1}, t_{q'-1}) = \text{cov}[\dot{y}_d^q(t_{q-1})\dot{y}_{d'}^{q'}(t_{q'-1})],$$
$$k_{m_d, f_{d'}}^q(t, t') = \text{cov}[m_d^q(t - t_{q-1})f_{d'}^q(t' - t_{q-1})],$$

and $\mathcal{X}_d^1$, $\mathcal{X}_d^2$, $\mathcal{X}_d^3$ and $\mathcal{X}_d^4$ are functions of the form $\sum_{n=2}^{q-q'} \prod_{i=2}^{q-q'} x_d^{q-i+1}(t_{q-i+1} - t_{q-i})$, with $x_d^{q-i+1}$ being equal to $c_d^{q-i+1}$, $e_d^{q-i+1}$, $g_d^{q-i+1}$ or $h_d^{q-i+1}$, depending on the values of $q$ and $q'$.

A similar expression to (4) can be obtained for $q' > q$. Examples of these functions for specific values of $q$ and $q'$ and more details are also given in the supplementary material.

## 4  Related work

There has been a recent interest in employing Gaussian processes for detection of change points in time series analysis, an area of study that relates to some extent to our model. Some machine learning related papers include [3, 4, 9]. [3, 4] deals specifically with how to construct covariance functions

in the presence of change points (see [3], section 4). The authors propose different alternatives according to the type of change point. From these alternatives, the closest ones to our work appear in subsections 4.2, 4.3 and 4.4. In subsection 4.2, a mechanism to keep continuity in a covariance function when there are two regimes described by different GPs, is proposed. The authors call this covariance continuous conditionally independent covariance function. In our switched latent force model, a more natural option is to use the initial conditions as the way to transit smoothly between different regimes. In subsections 4.3 and 4.4, the authors propose covariances that account for a sudden change in the input scale and a sudden change in the output scale. Both type of changes are automatically included in our model due to the latent force model construction: the changes in the input scale are accounted by the different length-scales of the latent force GP process and the changes in the output scale are accounted by the different sensitivity parameters. Importantly, we also concerned about multiple output systems.

On the other hand, [9] proposes an efficient inference procedure for Bayesian Online Change Point Detection (BOCPD) in which the underlying predictive model (UPM) is a GP. This reference is less concerned about the particular type of change that is represented by the model: in our application scenario, the continuity of the covariance function between two regimes must be assured beforehand.

## 5    Implementation

In this section, we describe additional details on the implementation, i.e., covariance function, hyperparameters, sparse approximations.

**Additional covariance functions.**    The covariance functions $k_{\dot{z}_d, z_{d'}}(t, t')$, $k_{z_d, \dot{z}_{d'}}(t, t')$ and $k_{\dot{z}_d, \dot{z}_{d'}}(t, t')$ are obtained by taking derivatives of $k_{z_d, z_{d'}}(t, t')$ with respect to $t$ and $t'$ [10].

**Estimation of hyperparameters.**    Given the number of outputs $D$ and the number of intervals $Q$, we estimate the parameters $\boldsymbol{\theta}$ by maximizing the marginal-likelihood of the joint Gaussian process $\{z_d(t)\}_{d=1}^D$ using gradient-descent methods. With a set of input points, $\mathbf{t} = \{t_n\}_{n=1}^N$, the marginal-likelihood is given as $p(\mathbf{z}|\boldsymbol{\theta}) = \mathcal{N}(\mathbf{z}|\mathbf{0}, \mathbf{K}_{\mathbf{z},\mathbf{z}} + \boldsymbol{\Sigma})$, where $\mathbf{z} = [\mathbf{z}_1^\top, \ldots, \mathbf{z}_D^\top]^\top$, with $\mathbf{z}_d = [z_d(t_1), \ldots, z_d(t_N)]^\top$, $\mathbf{K}_{\mathbf{z},\mathbf{z}}$ is a $D \times D$ block-partitioned matrix with blocks $\mathbf{K}_{\mathbf{z}_d, \mathbf{z}_{d'}}$. The entries in each of these blocks are evaluated using $k_{z_d, z_{d'}}(t, t')$. Furthermore, $k_{z_d, z_{d'}}(t, t')$ is computed using the expressions (3), and (4), according to the relative values of $q$ and $q'$.

**Efficient approximations**    Optimizing the marginal likelihood involves the inversion of the matrix $\mathbf{K}_{\mathbf{z},\mathbf{z}}$, inversion that grows with complexity $\mathcal{O}(D^3 N^3)$. We use a sparse approximation based on variational methods presented in [2] as a generalization of [11] for multiple output Gaussian processes. The approximations establish a lower bound on the marginal likelihood and reduce computational complexity to $\mathcal{O}(DNK^2)$, being $K$ a reduced number of points used to represent $u(t)$.

## 6    Experimental results

We now show results with artificial data and data recorded from a robot performing a basic set of actions appearing in table tennis.

### 6.1    Toy example

Using the model, we generate samples from the GP with covariance function as explained before. In the first experiment, we sample from a model with $D = 2$, $R = 1$ and $Q = 3$, with switching points $t_0 = -1, t_1 = 5$ and $t_2 = 12$. For the outputs, we have $A_1 = A_2 = 0.1$, $C_1 = 0.4, C_2 = 1$, $\kappa_1 = 2, \kappa_2 = 3$. We restrict the latent forces to have the same length-scale value $\ell_0 = \ell_1 = \ell_2 = 1e-3$, but change the values of the sensitivity parameters as $S_{1,1} = 10, S_{2,1} = 1, S_{1,2} = 10, S_{2,2} = 5, S_{1,3} = -10$ and $S_{2,3} = 1$, where the first subindex refers to the output $d$ and the second subindex refers to the force in the interval $q$. In this first experiment, we wanted to show the ability of the model to detect changes in the sensitivities of the forces, while keeping the length scales equal along the intervals. We sampled 5 times from the model with each output having 500 data points and add some noise with variance equal to ten percent of the variance of each sampled output. In each of the five repetitions, we took $N = 200$ data points for training and the remaining 300 for testing.

|   |      | $Q=1$ | $Q=2$ | $Q=3$ | $Q=4$ | $Q=5$ |
|---|------|-------|-------|-------|-------|-------|
| 1 | SMSE | 76.27±35.63 | 14.66±11.74 | **0.30±0.02** | **0.31±0.03** | **0.72±0.56** |
|   | MSLL | −0.98±0.46 | −1.79±0.26 | **−2.90±0.03** | **−2.87±0.04** | **−2.55±0.41** |
| 2 | SMSE | 7.27±6.88 | **1.08±0.05** | **1.10±0.05** | **1.06±0.05** | **1.10±0.09** |
|   | MSLL | −1.79±0.28 | **−2.26±0.02** | **−2.25±0.02** | **−2.27±0.03** | **−2.26±0.06** |

Table 1: Standarized mean square error (SMSE) and mean standardized log loss (MSLL) using different values of $Q$ for both toy examples. The figures for the SMSE must be multiplied by $10^{-2}$. See the text for details.

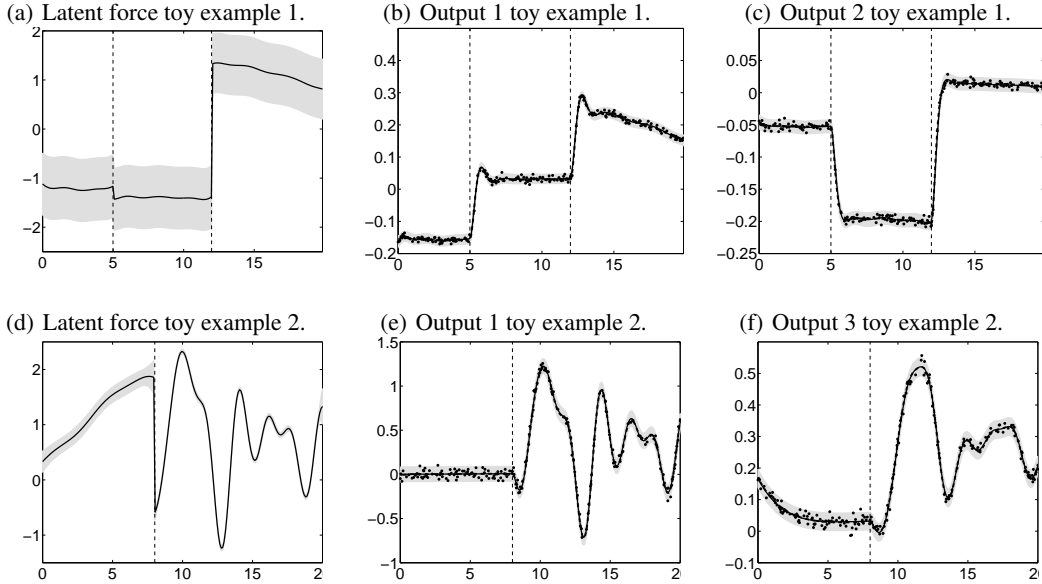

(a) Latent force toy example 1.  (b) Output 1 toy example 1.  (c) Output 2 toy example 1.

(d) Latent force toy example 2.  (e) Output 1 toy example 2.  (f) Output 3 toy example 2.

Figure 4: Mean and two standard deviations for the predictions over the latent force and two of the three outputs in the test set. Dashed lines indicate the final value of the swithcing points after optimization. Dots indicate training data.

Optimization of the hyperparameters (including $t_1$ and $t_2$) is done by maximization of the marginal likelihood through scaled conjugate gradient. We train models for $Q = 1, 2, 3, 4$ and $5$ and measure the mean standarized log loss (MSLL) and the mean standarized mean square error (SMSE) [8] over the test set for each value of $Q$. Table 1, first two rows, show the corresponding average results over the 5 repetitions together with one standard deviation. Notice that for $Q = 3$, the model gets by the first time the best performance, performance that repeats again for $Q = 4$. The SMSE performance remains approximately equal for values of $Q$ greater than 3. Figures 4(a), 4(b) and 4(c) shows the kind of predictions made by the model for $Q = 3$.

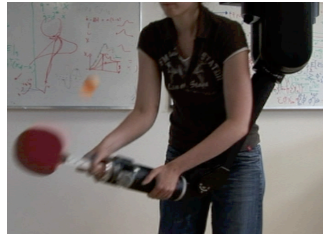

Figure 3: Data collection was performed using a Barrett WAM robot as haptic input device.

We generate also a different toy example, in which the length-scales of the intervals are different. For the second toy experiment, we assume $D = 3$, $Q = 2$ and switching points $t_0 = -2$ and $t_1 = 8$. The parameters of the outputs are $A_1 = A_2 = A_3 = 0.1$, $C_1 = 2, C_2 = 3, C_3 = 0.5$, $\kappa_1 = 0.4, \kappa_2 = 1, \kappa_3 = 1$ and length scales $\ell_0 = 1e-3$ and $\ell_1 = 1$. Sensitivities in this case are $S_{1,1} = 1, S_{2,1} = 5, S_{3,1} = 1, S_{1,2} = 5, S_{2,2} = 1$ and $S_{3,2} = 1$. We follow the same evaluation setup as in toy example 1. Table 1, last two rows, show the performance again in terms of MLSS and SMSE. We see that for values of $Q > 2$, the MLSS and SMSE remain similar. In figures 4(d), 4(e) and 4(f), the inferred latent force and the predictions made for two of the three outputs.

## 6.2 Segmentation of human movement data for robot imitation learning

In this section, we evaluate the feasibility of the model for motion segmentation with possible applications in the analysis of human movement data and imitation learning. To do so, we had a human teacher take the robot by the hand and have him demonstrate striking movements in a cooperative game of table tennis with another human being as shown in Figure 3. We recorded joint positions,

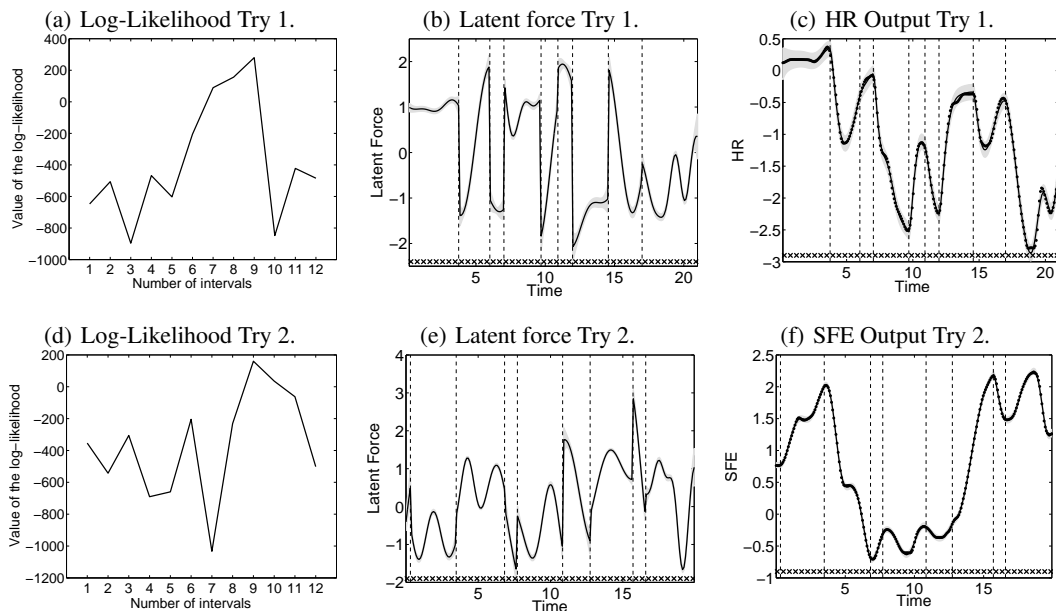

Figure 5: Employing the switching dynamical LFM model on the human movement data collected as in Fig.3 leads to plausible segmentations of the demonstrated trajectories. The first row corresponds to the log-likelihood, latent force and one of four outputs for trial one. Second row shows the same quantities for trial two. Crosses in the bottom of the figure refer to the number of points used for the approximation of the Gaussian process, in this case $K = 50$.

angular velocities, and angular acceleration of the robot for two independent trials of the same table tennis exercise. For each trial, we selected four output positions and train several models for different values of $Q$, including the latent force model without switches ($Q = 1$). We evaluate the quality of the segmentation in terms of the log-likelihood. Figure 5 shows the log-likelihood, the inferred latent force and one output for trial one (first row) and the corresponding quantities for trial two (second row). Figures 5(a) and 5(d) show peaks for the log-likelihood at $Q = 9$ for trial one and $Q = 10$ for trial two. As the movement has few gaps and the data has several output dimensions, it is hard even for a human being to detect the transitions between movements (unless it is visualized as in a movie). Nevertheless, the model found a maximum for the log-likelihood at the correct instances in time where the human transits between two movements. At these instances the human usually reacts due to an external stimulus with a large jerk causing a jump in the forces. As a result, we obtained not only a segmentation of the movement but also a generative model for table tennis striking movements.

## 7   Conclusion

We have introduced a new probabilistic model that develops the latent force modeling framework with switched Gaussian processes. This allows for discontinuities in the latent space of forces. We have shown the application of the model in toy examples and on a real world robot problem, in which we were interested in finding and representing striking movements. Other applications of the switching latent force model that we envisage include modeling human motion capture data using the second order ODE and a first order ODE for modeling of complex circuits in biological networks. To find the order of the model, this is, the number of intervals, we have used cross-validation. Future work includes proposing a less expensive model selection criteria.

**Acknowledgments**

MA and NL are very grateful for support from a Google Research Award "Mechanistically Inspired Convolution Processes for Learning" and the EPSRC Grant No EP/F005687/1 "Gaussian Processes for Systems Identification with Applications in Systems Biology". MA also thanks PASCAL2 Internal Visiting Programme. We also thank to three anonymous reviewers for their helpful comments.

**References**

[1] Mauricio Álvarez, David Luengo, and Neil D. Lawrence. Latent Force Models. In David van Dyk and Max Welling, editors, *Proceedings of the Twelfth International Conference on Artificial Intelligence and Statistics*, pages 9–16, Clearwater Beach, Florida, 16-18 April 2009. JMLR W&CP 5.

[2] Mauricio A. Álvarez, David Luengo, Michalis K. Titsias, and Neil D. Lawrence. Efficient multioutput Gaussian processes through variational inducing kernels. In *JMLR: W&CP 9*, pages 25–32, 2010.

[3] Roman Garnett, Michael A. Osborne, Steven Reece, Alex Rogers, and Stephen J. Roberts. Sequential Bayesian prediction in the presence of changepoints and faults. *The Computer Journal*, 2010. Advance Access published February 1, 2010.

[4] Roman Garnett, Michael A. Osborne, and Stephen J. Roberts. Sequential Bayesian prediction in the presence of changepoints. In *Proceedings of the 26th Annual International Conference on Machine Learning*, pages 345–352, 2009.

[5] Antti Honkela, Charles Girardot, E. Hilary Gustafson, Ya-Hsin Liu, Eileen E. M. Furlong, Neil D. Lawrence, and Magnus Rattray. Model-based method for transcription factor target identification with limited data. *PNAS*, 107(17):7793–7798, 2010.

[6] A. Ijspeert, J. Nakanishi, and S. Schaal. Learning attractor landscapes for learning motor primitives. In *Advances in Neural Information Processing Systems 15*, 2003.

[7] T. Oyama, Y. Uno, and S. Hosoe. Analysis of variability of human reaching movements based on the similarity preservation of arm trajectories. In *International Conference on Neural Information Processing (ICONIP)*, pages 923–932, 2007.

[8] Carl Edward Rasmussen and Christopher K. I. Williams. *Gaussian Processes for Machine Learning*. MIT Press, Cambridge, MA, 2006.

[9] Yunus Saatçi, Ryan Turner, and Carl Edward Rasmussen. Gaussian Process change point models. In *Proceedings of the 27th Annual International Conference on Machine Learning*, pages 927–934, 2010.

[10] E. Solak, R. Murray-Smith W. E. Leithead, D. J. Leith, and C. E. Rasmussen. Derivative observations in Gaussian process models of dynamic systems. In Sue Becker, Sebastian Thrun, and Klaus Obermayer, editors, *NIPS*, volume 15, pages 1033–1040, Cambridge, MA, 2003. MIT Press.

[11] Michalis K. Titsias. Variational learning of inducing variables in sparse Gaussian processes. In *JMLR: W&CP 5*, pages 567–574, 2009.

## Footnotes

[1] Derivations of these equations are rather involved. In the supplementary material, section 2, we include a detailed description of how to obtain the equations (3) and (4)

[2] See supplementary material, section 2.2.1.

[3] We will write $f_d^q(t - t_{q-1}, u_{q-1})$ as $f_d^q(t - t_{q-1})$ for notational simplicity.

[4] See supplementary material, section 2.2.2
